# Grouping Contours by Iterated Pairing Network

**Amnon Shashua**
M.I.T. Artificial Intelligence Lab., NE43-737
and Department of Brain and Cognitive Science
Cambridge, MA 02139

**Shimon Ullman**

## Abstract

We describe in this paper a network that performs grouping of image contours. The input to the net are fragments of image contours, and the output is the partitioning of the fragments into groups, together with a saliency measure for each group. The grouping is based on a measure of overall length and curvature. The network decomposes the overall optimization problem into independent optimal pairing problems performed at each node. The resulting computation maps into a uniform locally connected network of simple computing elements.

## 1 The Problem: Contour Grouping

A problem that often arises in visual information processing is the linking of contour fragments into optimal groups. For example, certain subsets of contours spontaneously form perceptual groups, as illustrated in Fig. 1, and are often detected immediately without scanning the image in a systematic manner. Grouping process of this type are likely to play an important role in object recognition by segmenting the image and selecting image structures that are likely to correspond to objects of interest in the scene.

We propose that some form of autonomous grouping is performed at an early stage based on geometrical characteristics, that are independent of the identity of objects to be selected. The grouping process is governed by the notion of saliency in a way that priority is given to forming salient groups at the expense of potentially less salient ones. This general notion can again be illustrated by Fig. 1; it appears that certain groups spontaneously emerge, while grouping decisions concerning the less salient parts of the image may remain unresolved. As we shall see, the computation below exhibits a similar behavior.

We define a grouping of the image contours as the formation of a set of disjoint

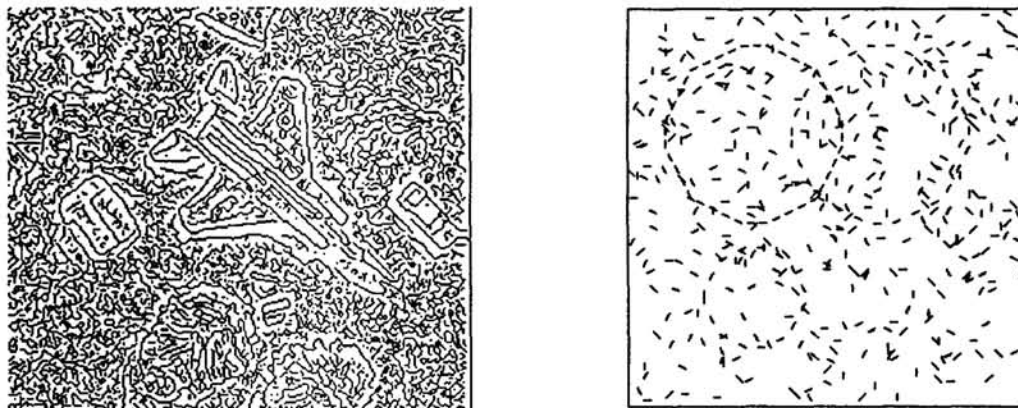

Figure 1: Contours that spontaneously form perceptual groups with various degrees of saliency. On the left is an edge image of a plane surrounded by a car, a house, trees and texture. The image on the right contains three circles, having decreasing degrees of saliency, in a background of randomly placed and oriented segments.

groups, each corresponding to a curve that may have any number of gaps, and whose union covers all the contour fragments in the image. Given a function $F(A)$ that measures some desired property of a group $A$, we would like to find a disjoint set of groups $\{A_1, ..., A_m\}$ that maximizes $\sum_i F(A_i)$ over all possible groupings. Our definition of the problem is related to, but not identical with, problems studied in the past under headings of "perceptual organization", "segmentation", "cueing" and "figure-ground separation". In our definition of grouping, local grouping decisions based on collinearity of neighboring edge segments may be overridden in favor of more global decisions that are governed by the overall saliency of the groups.

The paper introduces a novel grouping method having the following properties: (i) the grouping optimizes an overall saliency measure, (ii) the optimization problem is mapped onto a uniform locally connected network of simple computing elements, and (iii) the network's architecture and its computation are different in several respects from traditional neural network models.

## 2    Optimal Grouping

For the purpose of grouping it is convenient to consider the image as a graph of edge elements. The vertices of the graph correspond to image pixels, and the arcs to elementary edge fragments. The input to the grouping problem is a contour image, represented by a subset $E^r$ of the elements in the graph. A path in the graph corresponds to a contour in the image having any number of gaps. This implies that the grouping process implicitly bridges across gaps. This filling-in process is critical to any grouping scheme as demonstrated by the circles in Fig. 1.

The emphasis in this paper is on 1-D chains of elements such as objects' bounding contours. Grouping is therefore a collections of chains of $A_1, ..., A_m$ such that $A_i \cap A_j = \emptyset$  $i \neq j$ and $\cup_i A_i \supseteq E^r$. To define an optimal grouping we will define a function $F(A)$ that measures the quality of a group $A$. An optimal grouping is then a grouping that maximizes $\sum_{i=1}^m F(A_i)$ over all possible groupings of the elements.

## 2.1   The Quality Measure of a Group, $F(A)$

The definition of the measure $F(A)$ is motivated by both perceptual and computational considerations. In agreement with perceptual observations, it is defined to favor long smooth contours. Its form is also designed to facilitate distributed multistage optimization, as discussed below.

To define $F(A)$ of a chain of elements $A = \{e_1, ..., e_m\}$, consider first a single element $e_i$, and the $n$ preceding elements in the chain. We use first a quantity $s_n(i)$ which is the contribution of the $n$ preceding elements to $e_i$, which is:

$$s_n(i) = \sum_{j=\max\{1,i-n\}}^{i} \sigma_j C_{ij}$$

$\sigma_j$ is defined as 1 when $e_j$ corresponds to a contour fragment in the image and 0 for gaps. $s_n(i)$ is therefore simply a weighted sum of the contributions of the elements in the chain. The weighting factor $C_{ij}$ is taken to be a decreasing function of the total curvature of the path $\gamma_{ij}$ between elements $e_i$ and $e_j$. This will lead to a grouping that prefers curves with small overall curvature over wiggly ones. $C_{ij}$ is given by the formula:

$$C_{ij} = e^{-\int_{\gamma_{ij}} \left(\frac{d\theta}{ds}\right)^2 ds}$$

The exponent is the squared total curvature of the path between elements $e_i$ and $e_j$, and the resulting $C_{ij}$ lies between 0 (highly curved contour) and 1 (straight line). For a discrete sampling of the curve, $C_{ij}$ can be approximated by the product:

$$C_{ij} = \prod_{p=i}^{j+1} f_{p,p-1} \qquad C_{ii} = 1$$

where $f_{p,q}$ is referred to as the *coupling constant* between adjacent elements $e_p$ and $e_q$ and is given by $f_{p,q} = e^{-\alpha \tan \frac{\alpha}{2}}$ where $\alpha$ is the angle measuring the orientation difference between $p$ and $q$ [3]. In a similar manner, one can define $\bar{s}_n(i)$, the contribution of the $n$ elements following $e_i$ in the chain. $S_n(i) = s_n(i) + \bar{s}_n(i) - \sigma_i$ measures the contribution to element $e_i$ from both direction. This increases monotonically with the length and low total curvature of the curve passing through element $e_i$. Then then the overall quality of the chain $A$ is finally given by

$$F_n(A) = \sum_{i=1}^{m} S_n(i)$$

$F_n(A)$ increases quadratically with the size of $A$ and is non-linear with respect to the total curvature of $A$. Maximizing $\sum F(A_j)$ over all possible groupings will, therefore, prefer groups that are long and smooth. As $n$ increases, the measure $F_n$ will depend on larger portions of the curve surrounding each element, resulting in a finer discrimination between groups. In practice, we limit the measure to a finite $n$, and the optimal grouping is defined as:

$$I_n = arg \max_{m, A_1, ..., A_m} \sum_{i=1}^{m} F_n(A_i)$$

where the max is taken over all possible groupings. That is, we are looking for a grouping that will maximize the overall criterion function based on length and smoothness.

# 3   The Optimization Approach

Optimizing $I_n$ is a nonlinear problem with an energy landscape that can be quite complex making it difficult to find a global optimum, or even good local optima, using straightforward gradient descent methods. We define below a computation that proceeds in two stages, *saliency* and *pairing* stages, of $n$ steps each. In the saliency stage we compute, by iterating a local computation, optimal values of $S_n(i)$ for all elements in the graph. These values are an upper-bound on the saliency values achievable by any grouping. In the pairing stage we further update $S_n(i)$ by repeatedly forming local pairings of elements at each node of the graph. The details of both stages are given below.

## 3.1 Saliency Stage

For any given grouping $A_1, ..., A_m$, because they are disjoint, we have that

$$\sum_{j=1}^{m} F_n(A_j) = \sum_{i=1}^{N} S_n(i) \leq \sum \max_{\gamma_i} S_n(i)$$

where $N$ is the number of elements in the graph and $\gamma_i$ is a curve passing through element $e_i$. We denote $S_n(i)$ to be the *saliency* of element $e_i$ with respect to a curve $\gamma_i$. We therefore have that the maximal saliency value $S_n^*(i) = \max_{\gamma_i} S_n(i)$ is an upper-bound on the saliency value element $e_i$ receives on the optimal grouping $I_n$.

We define a local computation on the grid of elements such that each element $e_i$ computes maximal $S_n(i)$ by iterating the following simple computation, at each step taking the maximal contribution of its neighbors.

$$s_0(i) = \sigma_i$$
$$s_{n+1}(i) = \sigma_i + \max_j s_n(j) f_{ij} \tag{1}$$

where this computation is performed by all elements in parallel. It can be shown that at the n'th iteration $s_n(i)$ is maximal over all possible curves of length $n$, having any number of gaps, that come into $e_i$. Since $S_n(i) = s_n(i) + \bar{s}_n(i) - \sigma_i$, we have found the maximal $S_n(i)$ as well. For further details on the properties of this computation, see [3]. Note that since the computation is carried by all elements of the net, including gaps ($\sigma$ equals 0), the gaps are filled-in as a by-product of the computation. One can show that the filling-in contour between two end-elements has the smallest overall curvature, and therefore has the shape of a cubic spline.

## 3.2  Pairing Stage

Given the optimal saliency values $S_n^*(i)$ computed at the saliency stage we would like next to find a near-optimal grouping $I_n$. We first note the one-to-one correspondence between a grouping and a pairing of elements at each node of the graph. We define a pairing to be a partition of the $k$ elements around node $P$ into $\lceil \frac{k}{2} \rceil$ disjoint pairs. A pairing performed over all nodes of the net creates an equivalence relation over the elements of the net and therefore, by transitivity, determines a grouping. We therefore proceed by selecting a pairing at each node of the net that will yield a near optimal grouping $I_n$.

Given $s_n(i)$, the optimal saliency values computed by (1), and a pairing at node $P$

we update the saliency values by

$$s_{n+1}(i) = \sigma_i + s_n(j)f_{ij} \qquad (2)$$

where $e_i$ and $e_j$ are pairs determined by the pairing. This computation is exactly like (1) with the exception that (2) is applied to a fixed pairing while in (1) each element selects the neighbor with maximal contribution. Further applications of pairing followed by (2) allows the result of pairing decisions to propagate along curves and influence other pairing decisions. This gives rise to the notion of *iterated pairings*, a repetitive pairing procedure applied simultaneously over all nodes of the graph followed by saliency computation (2). We define below a pairing procedure that identifies salient groups in contour images.

For every node $P$ in the graph with elements $e_1, ..., e_k$ coming into $P$, we have that $s_n(i)$ $i = 1, ..., k$ computed by (1) are measured along optimal, not necessarily disjoint, curves $A_1, ..., A_k$ of length $n$ each. An *optimal pairing* at node $P$ is defined as a disjoint pairing that concatenates $A_1, ..., A_k$ into $\lceil \frac{k}{2} \rceil$ curves such that the sum of their quality measure $F(\cdot)$ is maximal. Because $F$ is defined to prefer smooth curves and because of its non-linearity with respect to total curvature, an optimal pairing agrees with the notion of forming salient groups on the expense of potentially less salient ones. The following proposition shows that an optimal pairing can be determined locally without the need to evaluate the quality measure of the concatenated curves.

**Proposition 1** *For a given node $P$, let $e_1, ..., e_k$ be the elements around $P$, $A_1, ..., A_k$ be curves coming into $P$ that are associated with the non-zero saliency values $s_1(n), ..., s_k(n)$ with sufficiently large $n$ (at least twice the largest chain $A_i$), $\pi$ be a permutation of the indices $(1, ..., k)$ and $J = \{(1,2), (3,4), ..., (k-1, k)\}$, then*

$$arg\max_\pi \sum_{(i,j) \in J} F_n(A_{\pi_i}, A_{\pi_j}) = arg\max_\pi \sum_{(i,j) \in J} \omega_{\pi_i, \pi_j}$$

*where $A_i A_j$ stands for the concatenation of curves $A_i, A_j$, and $\omega_{ij} = f_{ij}(s_n(i)c_n(j) + s_n(j)c_n(i))$ where $c_n$ is defined as $c_n(i) = \sum_k C_{kj}$ where $k$ is taken over all elements in the chain $A_j$.*

**Proof:** This is merely a calculation. $F_n(A_i A_j)$, the measure of group-saliency of the chain $A_i A_j$, is equal to $F_n(A_i) + F_n(A_j) + \omega_{ij}$. Finally, without loss of generality, we can assume that $k$ is even, because we can always add another element with zero weights attached to it. $\square$

Proposition 1 shows that an optimal pairing of elements can be determined locally on the basis of the saliency values computed in (1). One way to proceed is therefore the following. The quantities $c_n$ and therefore $\omega_{ij}$ can be accumulated and computed during computation (1). Then, the optimal pairing is computed at every node. Finding an optimal pairing is equivalent to finding an optimal *weighted match in a general graph* [2], with weights $\omega_{ij}$. The weighted matching problem on graphs has a polynomial algorithm due to Edmonds [1] and therefore its implementation is not unwieldly.

Below we describe an alternative and more biologically plausible scheme that can be implemented in a simple network using iterative local computations. The computation is in fact almost identical to the saliency computation described in (1).

Since the saliency values $s_n$ computed by (1) are an upper-bound on the final values achievable in any grouping, we would like to find a pairing that will preserve these values as closely as possible. Suppose that at $P$, $e_i$ receives its maximal contribution from $e_j$, and at the same time $e_i$ provides the maximal contribution to $e_j$ ('mutual neighbors'). When performing local pairing at $P$, it is reasonable to select $e_i$ and $e_j$ as a pair. Note that although this is a local decision at $P$, the values $s_n(i)$ and $s_n(j)$ already take into account the contribution of extended curves. The remaining elements undergo another round of saliency selection and pairing of mutual neighbors, until all elements at $P$ are paired. The following proposition shows that this pairing process is well behaved in the sense that at each selection round there will always be at least one pair that mutually select each other. We therefore have that the number of selection rounds is bounded by $\lceil \frac{k}{2} \rceil$, where $k$ is the number of elements having non-zero saliency value coming into node $P$.

**proposition 2** *Let $x_1, ..., x_k$ be $k$ positive real numbers, $\omega_{ij} = \omega_{ji}$ be positive weights $i, j = 1, ..., k$ and $\delta_i = arg \max_j x_j \omega_{ij}$, then $\exists i, j$ such that $\delta_i = j$ and $\delta_j = i$ (i and j are mutual neighbors).*

**Proof:** by induction on $k$. For $k = 3$ assume there exists a cycle in the selection pattern. For any given cycle we can renumber the indecis such that $\delta_1 = 2$, $\delta_2 = 3$ and $\delta_3 = 1$. Let $w_i$ stand for $w_{i-1,i}$ where $\omega_1 = \omega_{k,1}$. We get (i) $x_2\omega_2 > x_3\omega_1$, (ii) $x_2\omega_3 > x_1\omega_2$ and (iii) $x_1\omega_1 > x_2\omega_3$. From (ii) and (iii) we get an inequality that contradicts (i). For the induction hypothesis, assume the claim holds for arbitrary $k - 1$. We must show that the claim holds for $k$. Given the induction hypothesis we must show that there is no selection pattern that will give rise to a cycle of size $k$. Assume in contradiction that such a cycle exists. For any given cycle of size $k$ we can renumber the indecis such that $\delta_i = i + 1$ and $\delta_k = 1$ which implies that $x_i\omega_i > x_j\omega_{ij}$ for all $j \neq i$. In particular we have the following $k$ inequalities: $x_i\omega_i > x_{i-2}\omega_{i-1}$ where $i = 1, ..., k$. From the $k - 1$ inequalities corresponding to $i = 2, ..., k$ we get, by transitivity, that $x_1\omega_1 < x_{k-1}\omega_k$ which contradicts the remaining inequality that corresponds to $i = 1$. $\square$

### 3.3 Summary of Computation

The optimization is mapped onto a locally connected network with a simple uniform computation. The computation consists of the following steps. (i) Compute the saliency $S_n^*$ of each line element using the computation defined in (1). (ii) At each node perform a pairing of the line elements at the node. The pairing is performed by repeatedly selecting mutual neighbors. (iii) Update at each node the values $s_n$ based on the newly formed pairing (eq. 2). (iv) Go back to step 2.

These iterated pairings allow pairing decisions to propagate along maximally salient curves and influence other pairing decisions. In the implementation, the number of iterations $n$ is equal in both stages and as $n$ increases, the finer the pairing would be, resulting in a finer discrimination between groups. During the computation, the more salient groups emerge first, the less salient groups require additional iterations. Although the process is not guaranteed to converge to an optimal solution, it is a very simple computation that yields in practice good results. Some examples are shown in the next section.

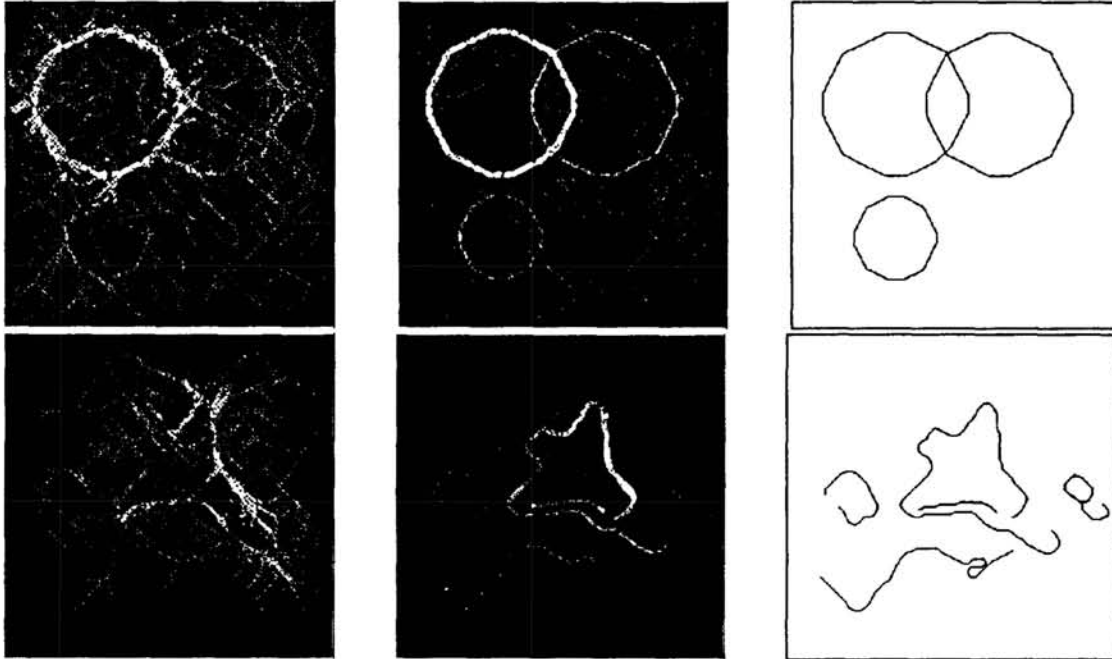

Figure 2: Results after 30 iterations of saliency and pairing on a net of size $128 \times 128$ with 16 elements per node. Images from left to right display the saliency map following the saliency and pairing stages and a number of strongest groups. The saliency of elements in the display is represented in terms of brightness and width — increased saliency measure corresponds to increase in brightness and width of element in display.

### 3.4 Examples

Fig. 2 shows the results of the network applied to the images in Fig. 1. The saliency values following the saliency and pairing stages illustrate that perceptually salient curves are also associated with high saliency values (see also [3]). Finally, in these examples, the highest saliency value of each group has been propagated along all elements of the group such that each group is now associated with a single saliency value. A number of strongest groups has been pulled out showing the close correspondence of these groups to objects of interest in the images.

### Acknowledgments

This work was supported by NSF grant IRI-8900267. Part of the work was done while A.S. was visiting the exploratory vision group at IBM research center, Yorktown Heights.

## References

[1] J. Edmonds. Path trees and flowers. *Can. J. Math.*, 1:263–271, 1965.

[2] C.H. Papadimitriou and K. Steiglitz. *Combinatorial Optimization: Algorithms and Complexity*. Prentice-Hall, New Jersey, 1982.

[3] A. Shashua and S. Ullman. Structural saliency: The detection of globally salient structures using a locally connected network. In *Proceedings of the 2nd International Conference on Computer Vision*, pages 321–327, 1988.